# Adaptive Template Matching with Shift-Invariant Semi-NMF

**Jonathan Le Roux**
Graduate School of Information
Science and Technology
The University of Tokyo
`leroux@hil.t.u-tokyo.ac.jp`

**Alain de Cheveigné**
CNRS, Université Paris 5,
and Ecole Normale Supérieure
`Alain.de.Cheveigne@ens.fr`

**Lucas C. Parra**[*]
Biomedical Engineering
City College of New York
City University of New York
`parra@ccny.cuny.edu`

## Abstract

How does one extract unknown but stereotypical events that are linearly superimposed within a signal with variable latencies and variable amplitudes? One could think of using template matching or matching pursuit to find the arbitrarily shifted linear components. However, traditional matching approaches require that the templates be known *a priori*. To overcome this restriction we use instead semi Non-Negative Matrix Factorization (semi-NMF) that we extend to allow for time shifts when matching the templates to the signal. The algorithm estimates templates directly from the data along with their non-negative amplitudes. The resulting method can be thought of as an adaptive template matching procedure. We demonstrate the procedure on the task of extracting spikes from single channel extracellular recordings. On these data the algorithm essentially performs spike detection and unsupervised spike clustering. Results on simulated data and extracellular recordings indicate that the method performs well for signal-to-noise ratios of 6dB or higher and that spike templates are recovered accurately provided they are sufficiently different.

## 1 Introduction

It is often the case that an observed waveform is the superposition of elementary waveforms, taken from a limited set and added with variable latencies and variable but positive amplitudes. Examples are a music waveform, made up of the superposition of stereotyped instrumental notes, or extracellular recordings of nerve activity, made up of the superposition of spikes from multiple neurons. In these examples, the elementary waveforms include both positive and negative excursions, but they usually contribute with a positive weight. Additionally, the elementary events are often temporally compact and their occurrence temporally sparse. Conventional template matching uses a known template and correlates it with the signal; events are assumed to occur at times where the correlation is high. Multiple template matching raises combinatorial issues that are addressed by Matching Pursuit [1]. However these techniques assume a preexisting dictionary of templates. We

---

[*]Corresponding author

wondered whether one can estimate the templates directly from the data, together with their timing and amplitude.

Over the last decade a number of blind decomposition methods have been developed that address a similar problem: given data, can one find the amplitudes and profiles of constituent signals that explain the data in some optimal way. This includes independent component analysis (ICA), non-negative matrix factorization (NMF), and a variety of other blind source separation algorithms. The different algorithms all assume a linear superposition of the templates, but vary in their specific assumptions about the statistics of the templates and the mixing process. These assumptions are necessary to obtain useful results because the problem is under-constrained.

ICA does not fit our needs because it does not implement the constraint that components (templates) are added with positive weights. NMF constrains weights to be non-negative but requires templates to also be non-negative. We will use instead the semi-NMF algorithm of Chris Ding [2, 3] that allows factoring a matrix into a product of a non-negative and an arbitrary matrix. To accommodate time shifts we modify it following the ideas of Morten Mørup [4] who presented a shift-invariant version of the NMF algorithm, that also includes sparsity constraints. We begin with the conventional formulation of the NMF modeling task as a matrix factorization problem and then derive in the subsequent section the case of a 1D sequence of data. NMF models a data matrix $X$ as a factorization,

$$\hat{X} = AB \,, \tag{1}$$

with $A \geq 0$ and $B \geq 0$ and finds these coefficients such that the square modeling error $||X - \hat{X}||^2$ is minimized. Matrix $A$ can be thought of as component amplitudes and the rows of matrix $B$ are the component templates. Semi-NMF drops the non-negative constraint for $B$, while shift-NMF allows the component templates to be shifted in time. In the NMF algorithm, there is an update equation for $A$ and an update equation for $B$. Semi-NMF and shift-NMF each modifies one of these equations, fortunately not the same, so their updates can be interleaved without interference.

## 2   Review of semi-NMF

Assume we are given $N$ observations or segments of data with $T$ samples arranged as a matrix $X_{nt}$. (The segments can also represent different epochs, trials, or even different channels.) The goal is to model this data as a linear superposition of $K$ component templates $B_{kt}$ with amplitudes $A_{nk}$, i.e.,

$$\hat{X}_{nt} = \sum_k A_{nk} B_{kt} = A_n^k B_{kt} \,. \tag{2}$$

The second expression here uses Einstein notation: indices that appear both as superscript and subscript within a product are to be summed. In contrast to matrix notation, all dimensions of an expression are apparent, including those that are absorbed by a sum, and the notation readily extends to more than two dimensions, which we will need when we introduce delays. We use this notation throughout the paper and include explicit sum signs only to avoid possible confusion.

Now, to minimize the modeling error

$$E = ||X - \hat{X}||_2^2 = \sum_{nt} \left( X_{nt} - A_n^k B_{kt} \right)^2 \,, \tag{3}$$

the semi-NMF algorithm iterates between finding the optimum $B$ for a given $A$, which is trivially given by the classic least squares solution,

$$B_{kt} = \left( A_k^n A_{k'n} \right)^{-1} A_n^{k'} X_t^n \,, \tag{4}$$

and improving the estimate of $A$ for a given $B$ with the multiplicative update

$$A_{nk} \leftarrow A_{nk} \sqrt{ \frac{ \left( X_n^t B_{kt} \right)^+ + A_n^{k'} \left( B_{k'}^t B_{kt} \right)^- }{ \left( X_n^t B_{kt} \right)^- + A_n^{k'} \left( B_{k'}^t B_{kt} \right)^+ } } \,. \tag{5}$$

In these expressions, $k'$ is a summation index; $(M)^{-1}$ stands for matrix inverse of $M$; and, $(M)^+ = \frac{1}{2}(|M| + M)$ and $(M)^- = \frac{1}{2}(|M| - M)$ are to be applied on each element of matrix $M$. The multiplicative update (5) ensures that $A$ remains non-negative in each step; while, baring constraints for $B$, the optimum solution for $B$ for a given $A$ is found in a single step with (4).

## 3  Shift-invariant semi-NMF

### 3.1  Formulation of the model for a 1D sequence

Consider now the case where the data is given as a 1-dimensional time sequence $X_t$. In the course of time, various events of unknown identity and variable amplitude appear in this signal. We describe an event of type $k$ with a template $B_{kl}$ of length $L$. Time index $l$ represents now a time lag measured from the onset of the template. An event can occur at any point in time, say at time sample $n$, and it may have a variable amplitude. In addition, we do not know *a priori* what the event type is and so we assign to each time sample $n$ and each event type $k$ an amplitude $A_{nk} \geq 0$. The goal is to find the templates $B$ and amplitudes $A$ that explain the data. In this formulation of the model, the timing of an event is given by a non-zero sample in the amplitude matrix $A$. Ideally, each event is identified uniquely and is well localized in time. This means that for a given $n$ the estimated amplitudes are positive for only one $k$, and neighboring samples in time have zero amplitudes. This new model can be written as

$$\hat{X}_t = \sum_n A_n^k B_{k,t-n} \tag{6}$$

$$= \sum_n \sum_l A_n^k \delta_{n,t-l} B_{kl} \,. \tag{7}$$

The Kronecker delta $\delta_{tl}$ was used to induce the desired shifts $n$. We can dispense with the cumbersome shift in the index if we introduce

$$\tilde{A}_{tkl} = \sum_n A_{nk} \delta_{n,t-l} \,. \tag{8}$$

The tensor $\tilde{A}_{tkl}$ represents a block Toeplitz matrix, with $K$ blocks of dimension $T \times L$. Each block implements a convolution of the $k$-th template $B_{kl}$ with amplitudes signal $A_{nk}$. With this definition the model is written now simply as:

$$\hat{X}_t = \tilde{A}_t^{kl} B_{kl} \,, \tag{9}$$

with $A_{nk} \geq 0$. We will also require a unit-norm constraint on the $K$ templates in $B$, namely, $B_k^l B_{kl} = 1$, to disambiguate the arbitrary scale in the product of $A$ and $B$.

### 3.2  Optimization criterion with sparseness prior

Under the assumption that the data represent a small set of well-localized events, matrix **A** should consist of a sparse series of pulses, the other samples having zero amplitude. To favor solutions having this property, we use a generalized Gaussian distribution as prior probability for the amplitudes. Assuming Gaussian white noise, the new cost function given by the negative log-posterior reads (up to a scaling factor),

$$E = \frac{1}{2}||X - \hat{X}||_2^2 + \beta\,||A||_\alpha^\alpha \tag{10}$$

$$= \frac{1}{2}\sum_t \left(X_t - \tilde{A}_t^{kl} B_{kl}\right)^2 + \beta \sum_{kl} A_{kl}^\alpha \,, \tag{11}$$

where $||\cdot||_p$ denotes the $L^p$ norm (or quasi-norm for $0 < p < 1$). The shape parameter $\alpha$ of the generalized Gaussian distribution controls the odds of observing low versus high amplitude values and should be chosen based on the expected rate of events. For our data we mostly choose $\alpha = 1/4$. The parameter $\beta$ is a normalization constant which depends on the power of the noise, $\sigma_N^2$, and the power of the amplitudes, $\sigma_A^2$, with $\beta = \frac{\sigma_N^2}{\sigma_A^\alpha}\left(\Gamma(3/\alpha)/\Gamma(1/\alpha)\right)^{\alpha/2}$.

### 3.3  $A$ update

The update for $A$ which minimizes this cost function is similar to update (5) with some modifications. In (5), amplitudes $A$ can be treated as a matrix of dimensions $T \times K$ and each update can be applied separately for every $n$. Here the problem is no longer separable in $n$ and we need to treat $A$ as a $1 \times TK$ matrix. $B$ is now a $TK \times T$ matrix of shifted templates defined as $\tilde{B}_{nkt} = B_{k,t-n}$. The new update equation is similar to (5), but differs in the term $BB^T$:

$$A_{nk} \leftarrow A_{nk} \sqrt{\frac{\left(\tilde{X}_n^l B_{kl}\right)^+ + A^{n'k'}\left(\tilde{B}_{n'k'}^t \tilde{B}_{nkt}\right)^-}{\left(\tilde{X}_n^l B_{kl}\right)^- + A^{n'k'}\left(\tilde{B}_{n'k'}^t \tilde{B}_{nkt}\right)^+ + \alpha\beta A_{nk}^{\alpha-1}}} . \qquad (12)$$

The summation in the $BB^T$ term is over $t$, and is 0 most of the time when the events do not overlap. We also defined $\tilde{X}_n^l = X_{n+l}$, and the time index in the summation $\tilde{X}_n^l B_{kl}$ extends only over lags $l$ from 0 to $L-1$. To limit the memory cost of this operation, we implemented it by computing only the non-zero parts of the $TK \times TK$ matrix $BB^T$ as $2L-1$ blocks of size $K \times K$. The extra term in the denominator of (12) is the gradient of the sparseness term in (11). A convergence proof for (12) can be obtained by modifying the convergence proof of the semi-NMF algorithm in [2] to include the extra $L_\alpha$ norm as penalty term. The proof relies on a new inequality on the $L_\alpha$ norm recently introduced by Kameoka to prove the convergence of his complex-NMF framework [5].

### 3.4  $B$ update

The templates $B$ that minimize the square modeling error, i.e., the first term of the cost function (11), are given by a least-squares solution which now writes:

$$B_{kl} = \left(\tilde{A}_{kl}^t \tilde{A}_{tk'l'}\right)^{-1} \tilde{A}_t^{k'l'} X^t . \qquad (13)$$

The matrix inverse is now over a matrix of $LK$ by $LK$ elements. Note that the sparseness prior will act to reduce the magnitude of $A$. Any scaling of $A$ can be compensated by a corresponding inverse scaling of $B$ so that the first term of the cost function remains unaffected. The unit-norm constraint for the templates $B$ therefore prevents $A$ from shrinking arbitrarily.

### 3.5  Normalization

The normalization constraint of the templates $B$ can be implemented using Lagrange multipliers, leading to the constrained least squares solution:

$$B_{kl} = \left(\tilde{A}_{kl}^t \tilde{A}_{tk'l'} + \Lambda_{kl,k'l'}\right)^{-1} \tilde{A}_t^{k'l'} X^t . \qquad (14)$$

Here, $\Lambda_{kl,k'l'}$ represents a diagonal matrix of size $KL \times KL$ with $K$ different Lagrange multipliers as parameters that need to be adjusted so that $B_k^l B_{kl} = 1$ for all $k$. This can be done with a Newton-Raphson root search of the $K$ functions $f_k(\Lambda) = B_k^l B_{kl} - 1$. The $K$ dimensional search for the Lagrange multipliers in $\Lambda$ can be interleaved with updates of $A$ and $B$. For simplicity however, in our first implementation we used the unconstrained least squares solution ($\Lambda = 0$) and renormalized $B$ and $A$ every 10 iterations.

## 4  Performance evaluations

We evaluated the algorithm on synthetic and real data. Synthetic data are used to provide a quantitative evaluation of performance as a function of SNR and the similarity of different templates. The algorithm is then applied to extracellular recordings of neuronal spiking activity and we evaluate its ability to recover two distinct spike types that are typically superimposed in this data.

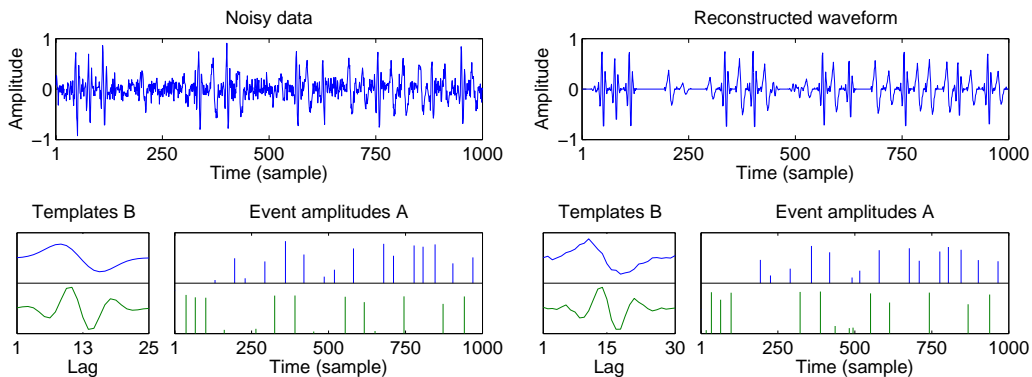

Figure 1: Example of synthetic spike trains and estimated model parameters at an SNR of 2 (6 dB). Top left: synthetic data. Bottom left: synthetic parameters (templates B and weight matrices A). Top right: reconstructed data. Bottom right: estimated parameters.

## 4.1 Quantitative evaluation on synthetic data

The goal of these simulations is to measure performance based on known truth data. We report detection rate, false-alarm rate, and classification error. In addition we report how accurately the templates have been recovered. We generated synthetic spike trains with two types of "spikes" and added Gaussian white noise. Figure 1 shows an example for SNR $= \sigma_A/\sigma_N = 2$ (or 6 dB). The two sets of panels show the templates $B$ (original on the left and recovered on the right), amplitudes $A$ (same as above) and noisy data $X$ (left) and estimated $\hat{X}$ (right). The figure shows the model parameters which resulted in a minimum cost. Clearly, for this SNR the templates have been recovered accurately and their occurrences within the waveform have been found with only a few missing events.

Performance as a function of varying SNR is shown in Figure 2. Detection rate is measured as the number of events recovered over the total number of events in the original data. False-alarms occur when noise is interpreted as actual events. Presence or absence of a recovered event is determined by comparing the original pulse train with the reconstructed pulse train $A$ (channel number $k$ is ignored). Templates in this example have a correlation time (3 dB down) of 2-4 samples and so we tolerate a misalignment of events of up to $\pm 2$ samples. We simulated 30 events with amplitudes uniformly distributed in $[0, 1]$. The algorithm tends to miss smaller events with amplitudes comparable to the noise amplitude. To capture this effect, we also report a detection rate that is weighted by event amplitude. Some events may be detected but assigned to the wrong template. We therefore report also classification performance. Finally, we report the goodness of fit as $R^2$ for the templates $B$ and the continuous valued amplitudes $A$ for the events that are present in the original data.

Note that the proposed algorithm implements implicitly a clustering and classification process. Obviously, the performance of this type of unsupervised clustering will degrade as the templates become more and more similar. Figure 2 shows the same performance numbers as a function of the similarity of the templates (without additive noise). A similarity of 0 corresponds to the templates shown as examples in Figure 1 (these are almost orthogonal with a cosine of 74°), and similarity 1 means identical templates. Evidently the algorithm is most reliable when the target templates are dissimilar.

## 4.2 Analysis of extracellular recordings

The original motivation for this algorithm was to analyze extracellular recordings from single electrodes in the guinea pig cochlear nucleus. Spherical and globular bushy cells in the anteroventral cochlear nucleus (AVCN) are assumed to function as reliable relays of spike trains from the auditory nerve, with "primary-like" responses that resemble those of auditory nerve fibers. Every incoming spike evokes a discharge within the outgoing axon [6].

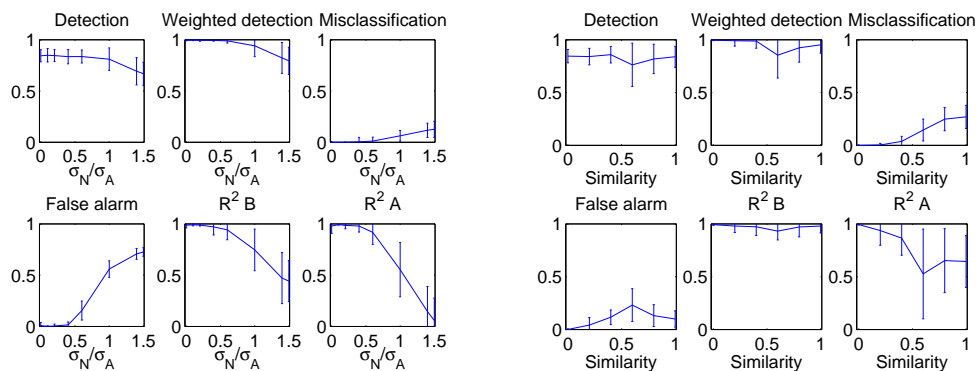

Figure 2: Left graph: performance as a function of SNR. Error bars represent standard deviation over 100 repetitions with varying random amplitudes and random noise. Top left: detection rate. Top center: weighted detection rate. Top right: misclassification rate (events attributed to the wrong template). Bottom left: false alarm rate (detected events which do not correspond to an event in the original data). Bottom center: $R^2$ of the templates $B$. Bottom right: $R^2$ of the amplitudes $A$. Right graph: same as a function of similarity between templates.

However, recent observations give a more nuanced picture, suggesting that the post-synaptic spike may sometimes be suppressed according to a process that is not well understood [7].

Extracellular recordings from primary-like cells within AVCN with a single electrode typically show a succession of events made up of three sub-events: a small pre-synaptic spike from the large auditory nerve fiber terminal, a medium-sized post-synaptic spike from the initial segment of the axon where it is triggered (the IS spike), and a large-sized spike produced by back-propagation into the soma and dendrites of the cell (the soma-dendritic or SD spike) (Fig. 3). Their relative amplitudes depend upon the position of the electrode tip relative to the cell. Our aim is to isolate each of these components to understand the process by which the SD spike is sometimes suppressed. The events may overlap in time (in particular the SD spike always overlaps with an IS spike), with varying positive amplitudes. They are temporally compact, on the order of a millisecond, and they occur repeatedly but sparsely throughout the recording, with positive amplitudes. The assumptions of our algorithm are met by these data, as well as by multi-unit recordings reflecting the activity of several neurons (the "spike sorting problem").

In the portions of our data that are sufficiently sparse (spontaneous activity), the components may be separated by an ad-hoc procedure: (a) trigger on the high-amplitude IS-soma complexes and set to zero, (b) trigger on the remaining isolated IS spikes and average to derive an IS spike template (the pre-synaptic spike is treated as part of the IS spike), (c) find the best match (in terms of regression) of the initial portion of the template to the initial portion of each IS-SD complex, (d) subtract the matching waveform to isolate the SD spikes, realign, and average to derive an SD spike template. The resulting templates are shown in Fig. 3 (top right). This ad-hoc procedure is highly dependent on prior assumptions, and we wished to have a more general and "agnostic" method to apply to a wider range of situations.

Figure 3 (bottom) shows the result of our automated algorithm. The automatically recovered spike templates seem to capture a number of the key features. Template 1, in blue, resembles the SD spike, and template 2, in red, is similar to the IS spike. The SD spikes are larger and have sharper peaks as compared to the IS spikes, while the IS spikes have an initial peak at 0.7 ms leading the main spike. The larger size of the extracted spikes corresponding to template 1 is correctly reflected in the histogram of the recovered amplitudes. However the estimated spike shapes are inaccurate. The main difference is in the small peak preceding the template 1. This is perhaps to be expected as the SD spike is always preceded in the raw data by a smaller IS spike. The expected templates were very similar (with a cosine of 38° as estimated from the manually extracted spikes), making the task particularly difficult.

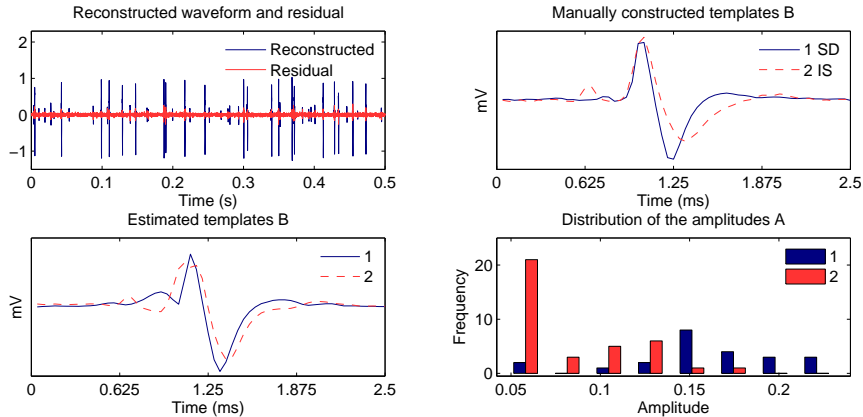

Figure 3: Experimental results on extracellular recordings. Top left: reconstructed waveform (blue) and residual between the original data and the reconstructed waveform (red). Top right: templates $B$ estimated manually from the data. Bottom left: estimated templates $B$. Bottom right: distribution of estimated amplitudes $A$. The SD spikes (blue) generally occur with larger amplitudes than the IS spikes (red).

## 4.3    Implementation details

As with the original NMF and semi-NMF algorithms, the present algorithm is only locally convergent. To obtain good solutions, we restart the algorithm several times with random initializations for $A$ (drawn independently from the uniform distribution in $[0, 1]$) and select the solution with the maximum posterior likelihood or minimum cost (11). In addition to these multiple restarts, we use a few heuristics that are motivated by the desired result of spike detection. We can thus prevent the algorithm from converging to some obviously suboptimal solutions:

**Re-centering the templates**: We noticed that local minima with poor performance typically occurred when the templates $B$ were not centered within the $L$ lags. In those cases the main peaks could be adjusted to fit the data, but the portion of the template that extends outside the window of $L$ samples could not be adjusted. To prune these suboptimal solutions, it was sufficient to center the templates during the updates while shifting the amplitudes accordingly.

**Pruning events**: We observed that spikes tended to generate non-zero amplitudes in $A$ in clusters of 1 to 3 samples. After convergence we compact these to pulses of 1-sample duration located at the center of these clusters. Spike amplitude was preserved by scaling the pulse amplitudes to match the sum of amplitudes in the cluster.

**Re-training with a less conservative sparseness constraint**: To ensure that templates $B$ are not affected by noise we initially train the algorithm with a strong penalty term (large $\beta$ effectively assuming strong noise power $\sigma_N^2$). Only spikes with large amplitudes remain after convergence and the templates are determined by only those strong spikes that have high SNR. After extracting templates accurately, we retrain the model amplitudes $A$ while keeping the templates $B$ fixed assuming now a weaker noise power (smaller $\beta$).

As a result of these steps, the algorithm converged frequently to good solutions (approximately 50 % of the time on the simulated data). The performance reported here represents the results with minimum error after 6 random restarts.

## 5    Discussion and outlook

**Alternative models**: The present 1D formulation of the problem is similar to that of Morten Mørup [4] who presented a 2D version of this model that is limited to non-negative templates. We have also derived a version of the model with observations $X$ arranged as a matrix, as well as a version in which event timing is encoded explicitly as time delays $\tau_n$

following [8]. We are omitting these alternative formulations here for the sake of brevity.

**Alternative priors**: In addition to the generalized Gaussian prior, we tested also Gaussian process priors [9] to encourage orthogonality between the $k$ sequences and refractoriness in time. However, we found that the quadratic expression of a Gaussian process competed with the $L_\alpha$ sparseness term. In the future, we intend to combine both criteria by allowing for correlations in the generalized Gaussian. The corresponding distributions are known as elliptically symmetric densities [10] and the corresponding process is called a spherically invariant random processes, e.g., [11].

**Sparseness and dimensionality reduction**: As with many linear decomposition methods, a key feature of the algorithm is to represent the data within a small linear subspace. This is particularly true for the semi-NMF algorithm since, provided a sufficiently large $K$ and without enforcing a sparsity constraint, the positivity constraint on $A$ actually amounts to no constraint at all (identical templates with opposite sign can accomplish the same as allowing negative $A$). For instance, without sparseness constraint on the amplitudes, a trivial solution in our examples above would be a template $B_{1l}$ with a single positive spike somewhere and another template $B_{2l}$ with a single negative spike, and all the time course encoded in $A_{n1}$ and $A_{n2}$.

**MISO identification**: The identifiability problem is compounded by the fact that the estimation of templates $B$ in this present formulation represents a multiple-input single-output (MISO) system identification problem. In the general case, MISO identification is known to be under-determined [12]. In the present case, the ambiguities of MISO identification may be limited due to the fact that we allow only for limited system length $L$ as compared to the number of samples $N$. Essentially, as the number of examples increases with increasing length of the signal $X$, the ambiguity in $B$ is reduced.

These issues will be adressed in future work.

# References

[1] S. Mallat and Z. Zhang, "Matching pursuit with time-frequency dictionnaries," *IEEE Trans. Signal Process.*, vol. 41, pp. 3397–3415, 1993.

[2] C. Ding, T. Li, and M. I. Jordan, "Convex and semi-nonnegative matrix factorization for clustering and low-dimension representation," Lawrence Berkeley National Laboratory, Tech. Rep. LBNL-60428, 2006.

[3] T. Li and C. Ding, "The relationships among various nonnegative matrix factorization methods for clustering," in *Proc. ICDM*, 2006, pp. 362–371.

[4] M. Mørup, M. N. Schmidt, and L. K. Hansen, "Shift invariant sparse coding of image and music data," Technical University of Denmark, Tech. Rep. IMM2008-04659, 2008.

[5] H. Kameoka, N. Ono, K. Kashino, and S. Sagayama, "Complex NMF: A new sparse representation for acoustic signals," in *Proc. ICASSP*, Apr. 2009.

[6] P. X. Joris, L. H. Carney, P. H. Smith, and T. C. T. Yin, "Enhancement of neural synchronization in the anteroventral cochlear nucleus. I. Responses to tones at the characteristic frequency," *J. Neurophysiol.*, vol. 71, pp. 1022–1036, 1994.

[7] S. Arkadiusz, M. Sayles, and I. M. Winter, "Spike waveforms in the anteroventral cochlear nucleus revisited," in *ARO midwinter meeting*, no. Abstract #678, 2008.

[8] M. Mørup, K. H. Madsen, and L. K. Hansen, "Shifted non-negative matrix factorization," in *Proc. MLSP*, 2007, pp. 139–144.

[9] C. E. Rasmussen and C. K. I. Williams, *Gaussian Processes for Machine Learning*, ser. Adaptive Computation and Machine Learning. Cambridge, MA: The MIT Press, Jan. 2006.

[10] K. Fang, S. Kotz, and K. Ng, *Symmetric Multivariate and Related Distributions*. London: Chapman and Hall, 1990.

[11] M. Rangaswamy, D. Weiner, and A. Oeztuerk, "Non-Gaussian random vector identification using spherically invariant random processes," *IEEE Trans. Aerospace and Electronic Systems*, vol. 29, no. 1, pp. 111–123, Jan. 1993.

[12] J. Benesty, J. Chen, and Y. Huang, *Microphone Array Signal Processing*. Berlin, Germany: Springer-Verlag, 2008.
